# Basis Selection For Wavelet Regression

**Kevin R. Wheeler**
Caelum Research Corporation
NASA Ames Research Center
Mail Stop 269-1
Moffett Field, CA 94035
kwheeler@mail.arc.nasa.gov

**Atam P. Dhawan**
College of Engineering
University of Toledo
2801 W. Bancroft Street
Toledo, OH 43606
adhawan@eng.utoledo.edu

## Abstract

A wavelet basis selection procedure is presented for wavelet regression. Both the basis and threshold are selected using cross-validation. The method includes the capability of incorporating prior knowledge on the smoothness (or shape of the basis functions) into the basis selection procedure. The results of the method are demonstrated using widely published sampled functions. The results of the method are contrasted with other basis function based methods.

## 1  INTRODUCTION

Wavelet regression is a technique which attempts to reduce noise in a sampled function corrupted with noise. This is done by thresholding the small wavelet decomposition coefficients which represent mostly noise. Most of the papers published on wavelet regression have concentrated on the threshold selection process. This paper focuses on the effect that different wavelet bases have on cross-validation based threshold selection, and the error in the final result. This paper also suggests how prior information may be incorporated into the basis selection process, and the effects of choosing a wrong prior. Both orthogonal and biorthogonal wavelet bases were explored.

Wavelet regression is performed in three steps. The first step is to apply a discrete wavelet transform to the sampled data to produce decomposition coefficients. Next a threshold is applied to the coefficients. Then an inverse discrete wavelet transform is applied to these modified coefficients.

The basis selection procedure is demonstrated to perform better than other wavelet regression methods even when the wrong prior on the space of the basis selections is specified.

This paper is broken into the following sections. The background section gives a brief summary of the mathematical requirements of the discrete wavelet transform. This section is followed by a methodology section which outlines the basis selection algorithms, and the process for obtaining the presented results. This is followed by a results section and then a conclusion.

## 2  BACKGROUND

### 2.1  DISCRETE WAVELET TRANSFORM

The Discrete Wavelet Transform (DWT) [Daubechies, 92] is implemented as a series of projections onto scaling functions in $L_2(\Re)$. The initial assumption is that the original data samples lie in the finest space $V_0$, which is spanned by the scaling function $\phi \in V_0$ such that the collection $\{\phi(x-l) \mid l \in \mathcal{Z}\}$ is a Riesz basis of $V_0$. The first level of the dyadic decomposition then consists of projecting the data samples onto scaling functions which have been dilated to be twice as wide as the original $\phi$. These span the coarser space $V_{-1} : \{\phi(2x - 2l) \mid l \in \mathcal{Z}\}$. The information that is lost going from the finer to coarser scale is retained in what is known as wavelet coefficients. Instead of taking the difference, the wavelet coefficients can be obtained via a projection operation onto the wavelet basis functions $\psi$ which span a space known as $W_0$. The projections are typically implemented using Quadrature Mirror Filters (QMF) which are implemented as Finite Impulse Response filters (FIR). The next level of decomposition is obtained by again doubling the scaling functions and projecting the first scaling decomposition coefficients onto these functions. The difference in information between this level and the last one is contained in the wavelet coefficients for this level. In general, the scaling functions for level $j$ and translation $m$ may be represented by: $\phi_j^m(t) = 2^{\frac{-j}{2}}\phi(2^{-j}t - m)$ where $t \in [0, 2^k - 1]$, $k \geq 1, 1 \leq j \leq k, 0 \leq m \leq 2^{k-j} - 1$.

#### 2.1.1  Orthogonal

An orthogonal wavelet decomposition is defined such that the difference space $W_j$ is the orthogonal complement of $V_j$ in $V_{j+1}$ : $W_0 \perp V_0$ which means that the projection of the wavelet functions onto the scaling functions on a level is zero: $\langle \psi, \phi(\cdot - l) \rangle = 0, \, l \in Z$

This results in the wavelet spaces $W_j$ with $j \in Z$ being all mutually orthogonal. The refinement relations for an orthogonal decomposition may be written as: $\phi(x) = 2 \sum_k h_k \phi(2x - k)$ and $\psi(x) = 2 \sum_k g_k \phi(2x - k)$.

#### 2.1.2  Biorthogonal

Symmetry is as an important property when the scaling functions are used as interpolatory functions. Most commonly used interpolatory functions are symmetric. It is well known in the subband filtering community that symmetry and exact reconstruction are incompatible if the same FIR filters are used for reconstruction and decomposition (except for the Haar filter) [Daubechies, 92]. If we are willing to

use different filters for the analysis and synthesis banks, then symmetry and exact reconstruction are possible using biorthogonal wavelets. Biorthogonal wavelets have dual scaling $\tilde{\phi}$ and dual wavelet $\tilde{\psi}$ functions. These generate a dual multiresolution analysis with subspaces $\tilde{V}_j$ and $\tilde{W}_j$ so that: $\tilde{V}_j \perp W_j$ and $V_j \perp \tilde{W}_j$ and the orthogonality conditions can now be written as:

$$\langle \tilde{\phi}, \psi(\cdot - l) \rangle = \langle \tilde{\psi}, \phi(\cdot - l) \rangle = 0$$

$$\langle \tilde{\phi}_{j,l}, \phi_{k,m} \rangle = \delta_{j-k}, \delta_{l-m} \text{ for } l, m, j, k \in Z$$

$$\langle \tilde{\psi}_{j,l}, \psi_{k,m} \rangle = \delta_{j-k}, \delta_{l-m} \text{ for } l, m, j, k \in Z$$

where $\delta_{j-k} = 1$ when $j = k$, and zero otherwise.

The refinement relations for biorthogonal wavelets can be written:

$$\phi(x) = 2 \sum_k h_k \phi(2x - k) \text{ and } \psi(x) = 2 \sum_k g_k \phi(2x - k)$$

$$\tilde{\phi}(x) = 2 \sum_k \tilde{h}_k \tilde{\phi}(2x - k) \text{ and } \tilde{\psi}(x) = 2 \sum_k \tilde{g}_k \tilde{\phi}(2x - k)$$

Basically, this means that the scaling functions at one level are composed of linear combinations of scaling functions at the next finer level. The wavelet functions at one level are also composed of linear combinations of the scaling functions at the next finer level.

## 2.2   LIFTING AND SECOND GENERATION WAVELETS

Swelden's lifting scheme [Sweldens, 95a] is a way to transform a biorthogonal wavelet decomposition obtained from low order filters to one that could be obtained from higher order filters (more FIR filter coefficients), without applying the longer filters and thus saving computations. This method can be used to increase the number of vanishing moments of the wavelet, or change the shape of the wavelet. This means that several different filters (i.e. sets of basis functions) may be applied with properties relevant to the problem domain in a manner more efficient than directly applying the filters individually. This is beneficial to performing a search over the space of admissible basis functions meeting the problem domain requirements.

Swelden's Second Generation Wavelets [Sweldens, 95b] are a result of applying lifting to simple interpolating biorthogonal wavelets, and redefining the refinement relation of the dual wavelet to be:

$$\tilde{\psi}(x) = \tilde{\phi}(2x - 1) - \sum_k a_k \tilde{\phi}(x - k)$$

where the $a_k$ are the lifting parameters. The lifting parameters may be selected to achieve desired properties in the basis functions relevant to the problem domain.

Prior information for a particular application domain may now be incorporated into the basis selection for wavelet regression. For example, if a particular application requires that there be a certain degree of smoothness (or a certain number of vanishing moments in the basis), then only those lifting parameters which result in a number of vanishing moments within this range are used. Another way to think

about this is to form a probability distribution over the space of lifting parameters. The most likely lifting parameters will be those which most closely match one's intuition for the given problem domain.

## 2.3   THRESHOLD SELECTION

Since the wavelet transform is a linear operator the decomposition coefficients will have the same form of noise as the sampled data. The idea behind wavelet regression is that the decomposition coefficients that have a small magnitude are substantially representative of the noise component of the sampled data. A threshold is selected and then all coefficients which are below the threshold in magntiude are either set to zero (a hard threshold) or a moved towards zero (a soft threshold). The soft threshold $\eta_t(y) = \text{sgn}(y)(\mid y \mid -t)$ is used in this study.

There are two basic methods of threshold selection: 1. Donoho's [Donoho, 95] analytic method which relies on knowledge of the noise distribution (such as a Gaussian noise source with a certain variance); 2. a cross-validation approach (many of which are reviewed in [Nason, 96]). It is beyond the scope of this paper to review these methods. Leave-one-out cross-validation with padding was used in this study.

## 3   METHODOLOGY

The test functions used in this study are the four functions published by Donoho and Johnstone [Donoho and Johnstone, 94]. These functions have been adopted by the wavelet regression community to aid in comparison of algorithms across publications.

Each function was uniformly sampled to contain 2048 points. Gaussian white noise was added so that the signal to noise ratio (SNR) was 7.0. Fifty replicates of each noisy function were created, of which four instantiations are depicted in Figure 1.

The noise removal process involved three steps. The first step was to perform a discrete wavelet transform using a paticular basis. A threshold was selected for the resulting decomposition coefficients using leave-one-out cross validation with padding.

The soft threshold was then applied to the decomposition. Next, the inverse wavelet transform was applied to obtain a cleaner version of the original signal. These steps were repeated for each basis set or for each set of lifting parameters.

## 3.1   WAVELET BASIS SELECTION

To demonstrate the effect of basis selection on the threshold found and the error in the resulting recovered signal, the following experiments were conducted. In the first trial two well studied orthogonal wavelet families were used: Daubechies most compactly supported (DMCS), and Symlets (S) [Daubechies, 92]. For the DMCS family, filters of order 1 (which corresponds to the Haar wavelet) through 7 were used. For the Symlets, filters of order 2 through 8 were used. For each filter, leave-one-out cross-validation was used to find a threshold which minimized the mean square error for each of the 50 replicates for the four test functions. The median threshold found was then applied to the decomposition of each of the replicates

for each test function. The resulting reconstructed signals are compared to the ideal function (the original before noise was added) and the Normalized Root Mean Square Error (NRMSE) is presented.

## 3.2   INCORPORATING PRIOR INFORMATION: LIFTING PARAMETERS

If the function that we are sampling is known to have certain smoothness properties, then a distribution of the admissible lifting coefficients representing a similar smoothness characteristic can be formed. However, it is not necessary to cautiously pick a prior. The performance of this method with a piecewise linear prior (the (2,2) biorthogonal wavelet of Cohen-Daubechies-Feauveau [Cohen, 92]) has been applied to the non-linear smooth test functions Bumps, Doppler, and Heavysin. This method has been compared with several standard techniques [Wheeler, 96]. The Smoothing Spline method (SS) [Wahba, 90], Donoho's Sure Shrink method (SureShrink)[Donoho, 95], and an optimized Radial Basis Function Neural Network (RBFNN).

## 4   RESULTS

In the first experiment, the procedure was only allowed to select between two well known bases (Daubechies most compactly supported and symmlet wavelets) with the desired filter order. Table 1 shows the filter order resulting in lowest cross-validation error for each filter and function. The NRMSE is presented with respect to the original noise-free functions for comparison. As expected the best basis for the noisy blocks function was the piecewise linear basis (Daubechies, order 1). The doppler, which had very high frequency components required the highest filter order. Figure 2 represents typical denoised versions for the functions recovered by the filters listed in **bold** in the table.

The method selected the basis having similar properties to the underlying function without knowing the original function. When higher order filters were applied to the noisy Blocks data, the resulting NRMSE was higher.

The basis selection procedure (labelled CV-Wavelets in Table 2) was compared with Donoho's SureShrink, Wahba's Smoothing Splines (SS), and an optimized RBFNN [Wheeler, 96]. The prior information specified incorrectly to the procedure to prefer bases near piecewise linear. The remarkable observation is that the method did better than the others as measured by Mean Square Error.

## 5   CONCLUSION

A basis selection procedure for wavelet regression was presented. The method was shown to select bases appropriate to the characteristics of the underlying functions. The shape of the basis was determined with cross-validation selecting from either a pre-set library of filters or from previously calculated lifting coefficients. The lifting coefficients were calculated to be appropriate for the particular problem domain. The method was compared for various bases and against other popular methods. Even with the wrong lifting parameters, the method was able to reduce error better than other standard algorithms.

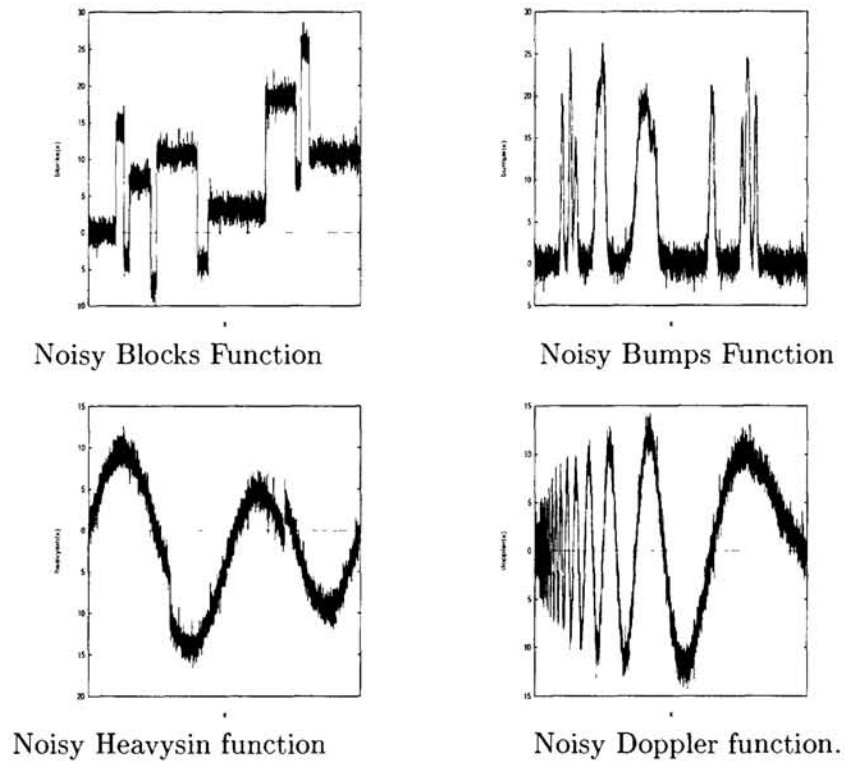

Noisy Blocks Function          Noisy Bumps Function

Noisy Heavysin function          Noisy Doppler function.

Figure 1: Noisy Test Functions

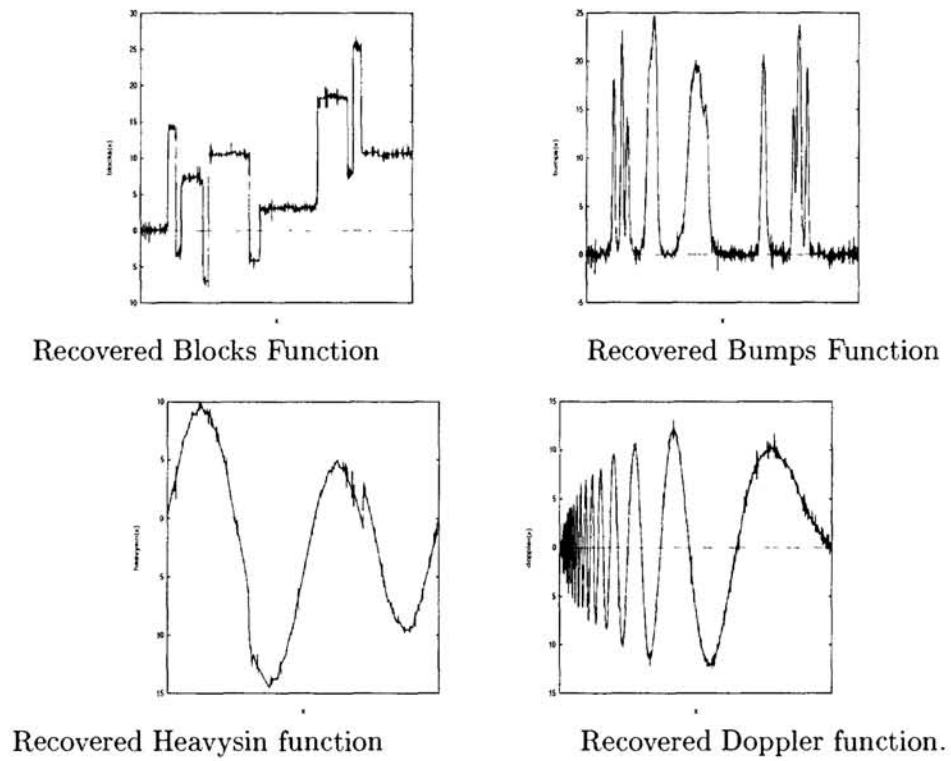

Recovered Blocks Function          Recovered Bumps Function

Recovered Heavysin function          Recovered Doppler function.

Figure 2: Recovered Functions

Table 1: Effects of Basis Selection

| Function | Filter Order | Family | Median Thr. (MT) | NRMSE Using MT | Median True Thr. | NRMSE using MTT |
|---|---|---|---|---|---|---|
| **Blocks** | **1** | **Daubechies** | **1.33** | **0.038** | **1.61** | **0.036** |
| Blocks | 2 | Symmlets | 1.245 | 0.045 | 1.40 | 0.045 |
| Bumps | 4 | Daubechies | 1.11 | 0.059 | 1.47 | 0.056 |
| **Bumps** | **5** | **Symmlets** | **1.13** | **0.058** | **1.48** | **0.055** |
| Doppler | 8 | Daubechies | 1.27 | 0.058 | 1.65 | 0.054 |
| **Doppler** | **8** | **Symmlets** | **1.36** | **0.054** | **1.74** | **0.050** |
| **Heavysin** | **2** | **Daubechies** | **1.97** | **0.039** | **2.17** | **0.038** |
| Heavysin | 5 | Symmlets | 1.985 | 0.039 | 2.16 | 0.038 |

Table 2: Methods Comparison Table of MSE

| Function | SS | SureShrink | RBFNN | CV-Wavelets |
|---|---|---|---|---|
| Blocks | 0.546 | 0.398 | 1.281 | 0.362 |
| Heavysin | 0.075 | 0.062 | 0.113 | 0.051 |
| Doppler | 0.205 | 0.145 | 0.287 | 0.116 |

## References

A. Cohen, I. Daubechies, and J. C. Feauveau (1992), "Biorthogonal bases of compactly supported wavelets," *Communications on Pure and Applied Mathematics*, vol. 45, no. 5, pp. 485 - 560, June.

I. Daubechies (1992), *Ten Lectures on Wavelets*, CBMS-NSF Regional Conference Series in Applied Mathematics, vol. 61, SIAM, Philadelphia, PA.

D. L. Donoho (1995), "De-noising by soft-thresholding," *IEEE Transactions on Information Theory*, vol. 41, no. 3, pp.613-627, May.

D. L. Donoho, I. M. Johnstone (1994), "Ideal spatial adaptation by wavelet shrinkage," *Biometrika*, vol. 81, no. 3, pp. 425-455, September.

G. P. Nason (1996), "Wavelet shrinkage using cross-validation," *Journal of the Royal Statistical Society, Series B*, vol. 58, pp. 463 - 479.

W. Sweldens (1995), "The lifting scheme: a custom-design construction of biorthogonal wavelets," Technical Report, no. IMI 1994:7, Dept. of Mathematics, University of South Carolina.

W. Sweldens (1995), "The lifting scheme: a construction of second generation wavelets," Technical Report, no. IMI 1995:6, Dept. of Mathematics, University of South Carolina.

G. Wahba (1990), *Spline Models for Observational Data*, SIAM, Philadelphia, PA.

K. Wheeler (1996), *Smoothing Non-uniform Data Samples With Wavelets*, Ph.D. Thesis, University of Cincinnati, Dept. of Electrical and Computer Engineering, Cincinnati, OH.